# Spectral Hashing

**Yair Weiss**[1,3]
[3]School of Computer Science,
Hebrew University,
91904, Jerusalem, Israel
yweiss@cs.huji.ac.il

**Antonio Torralba**[1]
[1]CSAIL, MIT,
32 Vassar St.,
Cambridge, MA 02139
torralba@csail.mit.edu

**Rob Fergus**[2]
[2]Courant Institute, NYU,
715 Broadway,
New York, NY 10003
fergus@cs.nyu.edu

## Abstract

Semantic hashing[1] seeks compact binary codes of data-points so that the Hamming distance between codewords correlates with semantic similarity. In this paper, we show that the problem of finding a best code for a given dataset is closely related to the problem of graph partitioning and can be shown to be NP hard. By relaxing the original problem, we obtain a spectral method whose solutions are simply a subset of thresholded eigenvectors of the graph Laplacian. By utilizing recent results on convergence of graph Laplacian eigenvectors to the Laplace-Beltrami eigenfunctions of manifolds, we show how to efficiently calculate the code of a novel datapoint. Taken together, both learning the code and applying it to a novel point are extremely simple. Our experiments show that our codes outperform the state-of-the art.

## 1   Introduction

With the advent of the Internet, it is now possible to use huge training sets to address challenging tasks in machine learning. As a motivating example, consider the recent work of Torralba et al. who collected a dataset of 80 million images from the Internet [2, 3]. They then used this weakly labeled dataset to perform scene categorization. To categorize a novel image, they simply searched for similar images in the dataset and used the labels of these retrieved images to predict the label of the novel image. A similar approach was used in [4] for scene completion.

Although conceptually simple, actually carrying out such methods requires highly efficient ways of (1) storing millions of images in memory and (2) quickly finding similar images to a target image.

Semantic hashing, introduced by Salakhutdinov and Hinton[5] , is a clever way of addressing both of these challenges. In semantic hashing, each item in the database is represented by a compact binary code. The code is constructed so that *similar items will have similar binary codewords* and there is a simple feedforward network that can calculate the binary code for a novel input. Retrieving similar neighbors is then done simply by retrieving all items with codes within a small Hamming distance of the code for the query. This kind of retrieval can be amazingly fast - millions of queries per second on standard computers. The key for this method to work is to learn a good code for the dataset. We need a code that is (1) easily computed for a novel input (2) requires a small number of bits to code the full dataset and (3) maps similar items to similar binary codewords.

To simplify the problem, we will assume that the items have already been embedded in a Euclidean space, say $R^d$, in which Euclidean distance correlates with the desired similarity. The problem of finding such a Euclidean embedding has been addressed in a large

number of machine learning algorithms (e.g. [6, 7]). In some cases, domain knowledge can be used to define a good embedding. For example, Torralba et al. [3] found that a 512 dimensional descriptor known as the GIST descriptor, gives an embedding where Euclidean distance induces a reasonable similarity function on the items. But simply having Euclidean embedding does not give us a fast retrieval mechanism.

If we forget about the requirement of having a small number of bits in the codewords, then it is easy to design a binary code so that items that are close in Euclidean space will map to similar binary codewords. This is the basis of the popular locality sensitive hashing method E2LSH [8]. As shown in[8], if every bit in the code is calculated by a random linear projection followed by a random threshold, then the Hamming distance between codewords will asymptotically approach the Euclidean distance between the items. But in practice this method can lead to very inefficient codes. Figure 1 illustrates the problem on a toy dataset of points uniformly sampled in a two dimensional rectangle. The figure plots the average precision at Hamming distance 1 using a E2LSH encoding. As the number of bits increases the precision improves (and approaches one with many bits), but the rate of convergence can be very slow.

Rather than using random projections to define the bits in a code, several authors have pursued machine learning approaches. In [5] the authors used an autoencoder with several hidden layers. The architecture can be thought of as a restricted Boltzmann machine (RBM) in which there are only connections between layers and not within layers. In order to learn 32 bits, the middle layer of the autoencoder has 32 hidden units, and noise was injected during training to encourage these bits to be as binary as possible. This method indeed gives codes that are much more compact than the E2LSH codes. In [9] they used multiple stacked RBMs to learn a non-linear mapping between input vector and code bits. Backpropagation using an Neighborhood Components Analysis (NCA) objective function was used to refine the weights in the network to preserve the neighborhood structure of the input space. Figure 1 shows that the RBM gives much better performance compared to random bits. A simpler machine learning algorithm (Boosting SSC) was pursued in [10] who used adaBoost to classify a pair of input items as similar or nonsimilar. Each weak learner was a decision stump, and the output of all the weak learners on a given output is a binary code. Figure 1 shows that this boosting procedure also works much better than E2LSH codes, although slightly worse than the RBMs[1].

The success of machine learning approaches over LSH is not limited to synthetic data. In [5], RBMs gave several orders of magnitude improvement over LSH in document retrieval tasks. In [3] both RBMs and Boosting were used to learn binary codes for a database of millions of images and were found to outperform LSH. Also, the retrieval speed using these short binary codes was found to be significantly faster than LSH (which was faster than other methods such as KD trees).

The success of machine learning methods leads us to ask: what is the best code for performing semantic hashing for a given dataset? We formalize the requirements for a good code and show that these are equivalent to a particular form of graph partitioning. This shows that even for a single bit, the problem of finding optimal codes is NP hard. On the other hand, the analogy to graph partitioning suggests a relaxed version of the problem that leads to very efficient eigenvector solutions. These eigenvectors are exactly the eigenvectors used in many spectral algorithms including spectral clustering and Laplacian eigenmaps [6, 11]. This leads to a new algorithm, which we call "spectral hashing" where the bits are calculated by thresholding a subset of eigenvectors of the Laplacian of the similarity graph. By utilizing recent results on convergence of graph Laplacian eigenvectors to the Laplace-Beltrami eigenfunctions of manifolds, we show how to efficiently calculate the code of a novel datapoint. Taken together, both learning the code and applying it to a novel point are extremely simple. Our experiments show that our codes outperform the state-of-the art.

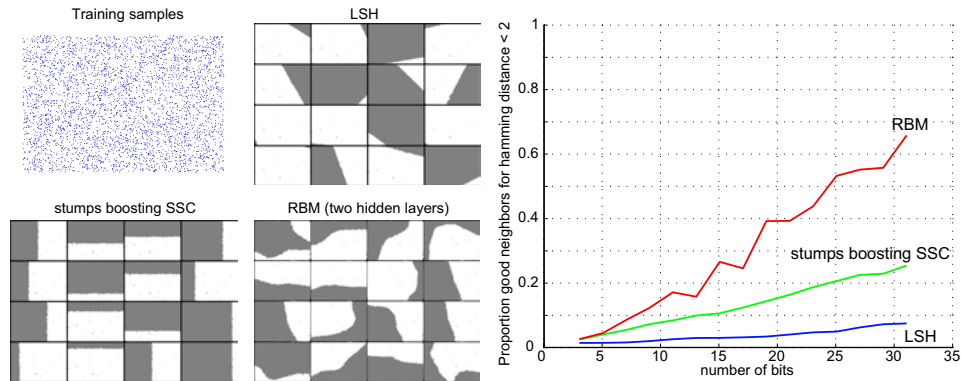

Figure 1: Building hash codes to find neighbors. Neighbors are defined as pairs of points in 2D whose Euclidean distance is less than $\epsilon$. The toy dataset is formed by uniformly sampling points in a two dimensional rectangle. The figure plots the average precision (number of neighbors in the original space divided by number of neighbors in a hamming ball using the hash codes) at Hamming distance $\leq 1$ for three methods. The plots on the left show how each method partitions the space to compute the bits to represent each sample. Despite the simplicity of this toy data, the methods still require many bits in order to get good performance.

## 2   Analysis: what makes a good code

As mentioned earlier, we seek a code that is (1) easily computed for a novel input (2) requires a small number of bits to code the full dataset and (3) maps similar items to similar binary codewords. Let us first ignore the first requirement, that codewords be easily computed for a novel input and search only for a code that is efficient (i.e. requires a small number of bits) and similarity preserving (i.e. maps similar items to similar codewords). For a code to be efficient, we require that each bit has a 50% chance of being one or zero, and that different bits are independent of each other. Among all codes that have this property, we will seek the ones where the average Hamming distance between similar points is minimal.

Let $\{y_i\}_{i=1}^n$ be the list of codewords (binary vectors of length $k$) for $n$ datapoints and $W_{n \times n}$ be the affinity matrix. Since we are assuming the inputs are embedded in $R^d$ so that Euclidean distance correlates with similarity, we will use $W(i,j) = \exp(-\|x_i - x_j\|^2/\epsilon^2)$. Thus the parameter $\epsilon$ defines the distance in $R^d$ which corresponds to similar items. Using this notation, the average Hamming distance between similar neighbors can be written: $\sum_{ij} W_{ij}\|y_i - y_j\|^2$. If we relax the independence assumption and require the bits to be *uncorrelated* we obtain the following problem:

$$minimize : \sum_{ij} W_{ij}\|y_i - y_j\|^2 \tag{1}$$

$$subject\ to : y_i \in \{-1, 1\}^k$$

$$\sum_i y_i = 0$$

$$\frac{1}{n}\sum_i y_i y_i^T = I$$

where the constraint $\sum_i y_i = 0$ requires each bit to fire 50% of the time, and the constraint $\frac{1}{n}\sum_i y_i y_i^T = I$ requires the bits to be uncorrelated.

**Observation:** For a single bit, solving problem 1 is equivalent to balanced graph partitioning and is NP hard.

**Proof:** Consider an undirected graph whose vertices are the datapoints and where the weight between item $i$ and $j$ is given by $W(i, j)$. Consider a code with a single bit. The bit partitions the graph into two equal parts $(A, B)$, vertices where the bit is on and vertices where the bit is off. For a single bit, $\sum_{ij} W_{ij} \|y_i - y_j\|^2$ is simply the weight of the edges cut by the partition: $cut(A, B) = \sum_{i \in A, j \in B} W(i, j)$. Thus problem 1 is equivalent to minimizing $cut(A, B)$ with the requirement that $|A| = |B|$ which is known to be NP hard [12].

For $k$ bits the problem can be thought of as trying to find $k$ independent balanced partitions, each of which should have as low cut as possible.

## 2.1 Spectral Relaxation

By introducing a $n \times k$ matrix $Y$ whose $j$th row is $y_j^T$ and a diagonal $n \times n$ matrix $D(i, i) = \sum_j W(i, j)$ we can rewrite the problem as:

$$minimize : trace(Y^T(D - W)Y) \tag{2}$$
$$subject\ to : Y(i, j) \in \{-1, 1\}$$
$$Y^T 1 = 0$$
$$Y^T Y = I$$

This is of course still a hard problem, but by removing the constraint that $Y(i, j) \in \{-1, 1\}$ we obtain an easy problem whose solutions are simply the $k$ eigenvectors of $D - W$ with minimal eigenvalue (after excluding the trivial eigenvector $1$ which has eigenvalue 0).

## 2.2 Out of Sample Extension

The fact that the solution to the relaxed problem are the $k$ eigenvectors of $D - W$ with minimal eigenvalue would suggest simply thresholding these eigenvectors to obtain a binary code. But this would only tell us how to compute the code representation of items in the training set. This is the problem of out-of-sample extension of spectral methods which is often solved using the Nystrom method [13, 14]. But note that the cost of calculating the Nystrom extension of a new datapoint is *linear* in the size of the dataset. In our setting, where there can be millions of items in the dataset this is impractical. In fact, calculating the Nystrom extension is as expensive as doing exhaustive nearest neighbor search.

In order to enable efficient out-of-sample extension we assume the datapoints $x_i \in R^d$ are samples from a probability distribution $p(x)$. The equations in the problem 1 are now seen to be sample averages which we replace with their expectations:

$$minimize : \int \|y(x_1) - y(x_2)\|^2 W(x_1, x_2) p(x_1) p(x_2) dx_1 x_2 \tag{3}$$
$$subject\ to : y(x) \in \{-1, 1\}^k$$
$$\int y(x) p(x) dx = 0$$
$$\int y(x) y(x)^T p(x) dx = I$$

with $W(x_1, x_2) = e^{-\|x_1 - x_2\|^2 / \epsilon^2}$. Relaxing the constraint that $y(x) \in \{-1, 1\}^k$ now gives a spectral problem whose solutions are *eigenfunctions* of the weighted Laplace-Beltrami operators defined on manifolds [15, 16, 13, 17]. More explicitly, define the weighted Laplacian $L_p$ as an operator that maps a function $f$ to $g = L_p f$ by $\frac{g(x)}{p(x)} = D(x)f(x)p(x) - \int_s W(s, x)f(s)p(s)ds$ with $D(x) = \int_s W(x, s)$. The solution to the relaxation of problem 3 are functions that satisfy $L_p f = \lambda f$ with minimal eigenvalue (ignoring the trivial solution $f(x) = 1$ which has eigenvalue 0). As discussed in [16, 15, 13], with proper normalization, the eigenvectors of the discrete Laplacian defined by $n$ points sampled from $p(x)$ converges to eigenfunctions of $L_p$ as $n \to \infty$.

What do the eigenfunctions of $L_p$ look like ? One important special case is when $p(x)$ is a separable distribution. A simple case of a separable distribution is a multidimensional

uniform distribution $\Pr(x) = \prod_i u_i(x_i)$ where $u_i$ is a uniform distribution in the range $[a_i, b_i]$. Another example is a multidimensional Gaussian, which is separable once the space has been rotated so that the Gaussian is axes aligned.

**Observation:** [17] If $p(x)$ is separable, and similarity between datapoints is defined as $e^{-\|x_i - x_j\|^2 / \epsilon^2}$ then the eigenfunctions of the continuous weighted Laplacian, $L_p$ have an outer product form. That is, if $\Phi_i(x)$ is an eigenfunction of the weighted Laplacian defined on $R^1$ with eigenvalue $\lambda_i$ then $\Phi_i(x_1)\Phi_j(x_2)\cdots\Phi_d(x_d)$ is an eigenfunction of the $d$ dimensional problem with eigenvalue $\lambda_i \lambda_j \cdots \lambda_d$.

Specifically for a case of a uniform distribution on $[a, b]$ the eigenfunctions of the one-dimensional Laplacian $L_p$ are extremely well studied objects in mathematics. They correspond to the fundamental modes of vibration of a metallic plate. The eigenfunctions $\Phi_k(x)$ and eigenvalues $\lambda_k$ are:

$$\Phi_k(x) \;=\; \sin(\frac{\pi}{2} + \frac{k\pi}{b-a}x) \tag{4}$$

$$\lambda_k \;=\; 1 - e^{-\frac{\epsilon^2}{2}|\frac{k\pi}{b-a}|^2} \tag{5}$$

A similar equation is also available for the one dimensional Gaussian . In this case the eigenfunctions of the one-dimensional Laplacian $L_p$ are (in the limit of small $\epsilon$) solutions to the Schrodinger equations and are related to Hermite polynomials. Figure 2 shows the analytical eigenfunctions for a 2D rectangle in order of increasing eigenvalue. The eigenvalue (which corresponds to the cut) determines which $k$ bits will be used. Note that the eigenvalue depends on the aspect ratio of the rectangle and the spatial frequency — it is better to cut the long dimension before the short one, and low spatial frequencies are preferred. Note that the eigenfunctions do not depend on the radius of similar neighbors $\epsilon$. The radius does change the eigenvalue but does not affect the ordering.

We distinguish between *single-dimension* eigenfunctions, which are of the form $\Phi_k(x_1)$ or $\Phi_k(x_2)$ and *outer-product* eigenfunctions which are of the form $\Phi_k(x_1)\Phi_l(x_2)$. These outer-product eigenfunctions are shown marked with a red border in the figure. As we now discuss, these outer-product eigenfunctions should be avoided when building a hashing code.

**Observation:** Suppose we build a code by thresholding the $k$ eigenfunctions of $L_p$ with minimal eigenvalue $y(x) = sign(\Phi_k(x))$. If any of the eigenfunctions is an outer-product eigenfunction, then that bit is a deterministic function of other bits in the code.

**Proof:** This follows from the fact that $sign(\Phi_1(x_1)\Phi_2(x_2)) = sign(\Phi_1(x_1))sign(\Phi_2(x_2))$.

This observation highlights the simplification we made in relaxing the independence constraint and requiring that the bits be uncorrelated. Indeed the bits corresponding to outer-product eigenfunctions are approximately uncorrelated but they are surely not independent.

The exact form of the eigenfunctions for 1D continuous Laplacian for different distributions is a matter of ongoing research [17]. We have found, however, that the bit codes obtained by thresholding the eigenfunctions are robust to the exact form of the distribution. In particular, simply fitting a multidimensional rectangle distribution to the data (by using PCA to align the axes, and then assuming a uniform distribution on each axis) works surprisingly well for a wide range of distributions. In particular, using the analytic eigenfunctions of a uniform distribution on data sampled from a Gaussian, works as well as using the numerically calculated eigenvectors and far better than boosting or RBMs trained on the Gaussian distribution.

To summarize, given a training set of points $\{x_i\}$ and a desired number of bits $k$ the **spectral hashing algorithm** works by:

- Finding the principal components of the data using PCA.
- Calculating the $k$ smallest *single-dimension* analytical eigenfunctions of $L_p$ using a rectangular approximation along every PCA direction. This is done by evaluating the $k$ smallest eigenvalues for each direction using (equation 4), thus creating a list of $dk$ eigenvalues, and then sorting this list to find the $k$ smallest eigenvalues.

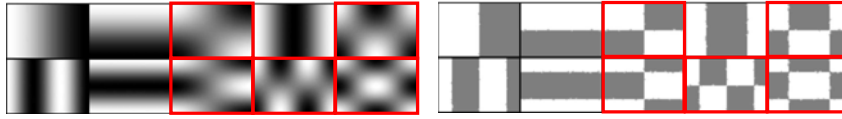

Figure 2: **Left:** Eigenfunctions for a uniform rectangular distribution in 2D. **Right:** Thresholded eigenfunctions. Outer-product eigenfunctions have a red frame. The eigenvalues depend on the aspect ratio of the rectangle and the spatial frequency of the cut – it is better to cut the long dimension first and lower spatial frequencies are better than higher ones.

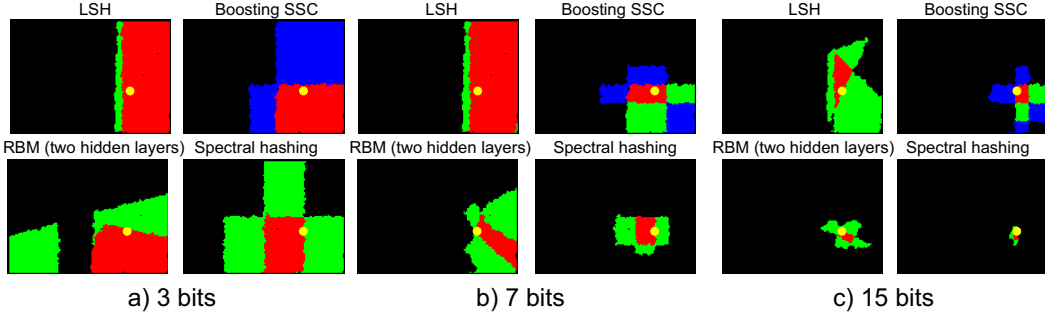

Figure 3: Comparison of neighborhood defined by hamming balls of different radii using codes obtained with LSH, Boosting, RBM and spectral hashing when using 3, 7 and 15 bits. The yellow dot denotes a test sample. The red points correspond to the locations that are within a hamming distance of zero. Green corresponds to a hamming ball of radius 1, and blue to radius 2.

- Thresholding the analytical eigenfunctions at zero, to obtain binary codes.

This simple algorithm has two obvious limitations. First, it assumes a multidimensional uniform distribution generated the data. We have experimented with using multidimensional Gaussians instead. Second, even though it avoids the trivial 3 way dependencies that arise from outer-product eigenfunctions, other high-order dependencies between the bits may exist. We have experimented with using only frequencies that are powers of two to avoid these dependencies. Neither of these more complicated variants of spectral hashing gave a significant improvement in performance in our experiments.

Figure 4a compares the performance of spectral hashing to LSH, RBMs and Boosting on a 2D rectangle and figure 3 visualizes the Hamming balls for the different methods. Despite the simplicity of spectral hashing, it outperforms the other methods. Even when we apply RBMs and Boosting to the output of spectral hashing the performance does not improve. A similar pattern of results is shown in high dimensional synthetic data (figure 4b).

Some insight into the superior performance can be obtained by comparing the partitions that each bit defines on the data (figures 2,1). Recall that we seek partitions that give low cut value and are approximately independent. LSH which uses random linear partitions may give very unbalanced partitions. RBMs and Boosting both find good partitions, but the partitions can be highly dependent on each other.

## 3   Results

In addition to the synthetic results we applied the different algorithms to the image databases discussed in [3]. Figure 5 shows retrieval results for spectral hashing, RBMs and boosting on the "labelme" dataset. Note that even though the spectral hashing uses a terrible model of the statistics of the database — it simply assumes a N dimensional rectangle, it performs better than boosting which actually uses the distribution (the difference in performance relative to RBMs is not significant). Not only is the performance numerically better, but

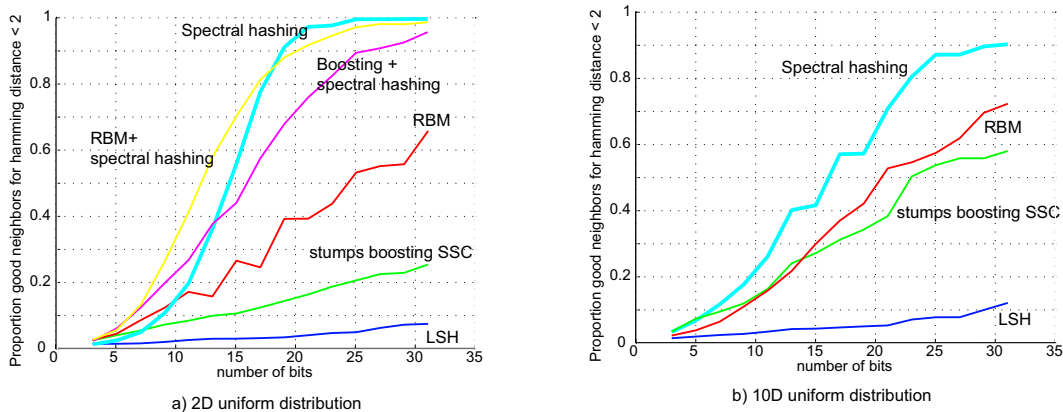

a) 2D uniform distribution

b) 10D uniform distribution

Figure 4: **left:** results on 2D rectangles with different methods. Even though spectral hashing is the simplest, it gives the best performance. **right:** Similar pattern of results for a 10 dimensional distribution.

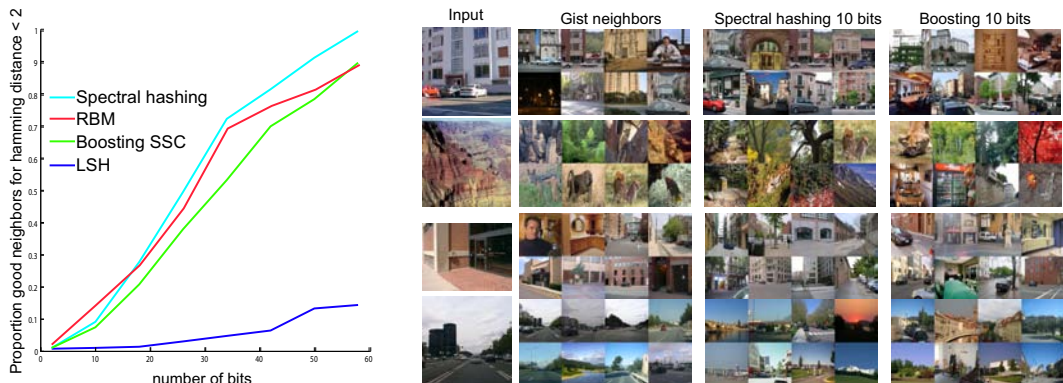

Figure 5: Performance of different binary codes on the LabelMe dataset described in [3]. The data is certainly not uniformly distributed, and yet spectral hashing gives better retrieval performance than boosting and LSH.

our visual inspection of the retrieved neighbors suggests that with a small number of bits, the retrieved images are better using spectral hashing than with boosting.

Figure 6 shows retrieval results on a dataset of 80 million images. This dataset is obviously more challenging and even using exhaustive search some of the retrieved neighbors are semantically quite different. Still, the majority of retrieved neighbors seem to be semantically relevant, and with 64 bits spectral hashing enables this peformance in fractions of a second.

# 4    Discussion

We have discussed the problem of learning a code for semantic hashing. We defined a hard criterion for a good code that is related to graph partitioning and used a spectral relaxation to obtain an eigenvector solution. We used recent results on convergence of graph Laplacian eigenvectors to obtain analytic solutions for certain distributions and showed the importance of avoiding redundant bits that arise from separable distributions.

The final algorithm we arrive at, spectral hashing, is extremely simple - one simply performs PCA on the data and then fits a multidimensional rectangle. The aspect ratio of this multidimensional rectangle determines the code using a simple formula. Despite this simplicity, the method is comparable, if not superior, to state-of-the-art methods.

| Gist neighbors | Spectral hashing: 32 bits | 64 bits |
| --- | --- | --- |

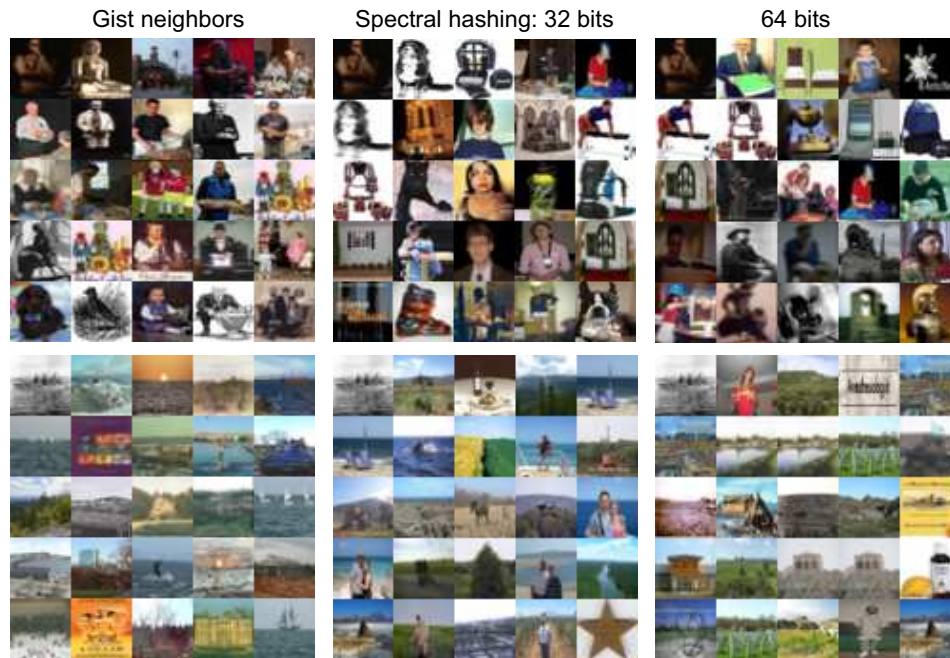

Figure 6: Retrieval results on a dataset of 80 million images using the original gist descriptor, and hash codes build with spectral hashing with 32 bits and 64 bits. The input image corresponds to the image on the top-left corner, the rest are the 24 nearest neighbors using hamming distance for the hash codes and L2 for gist.

## Footnotes

[1]All methods here use the same retrieval algorithm, i.e. semantic hashing. In many applications of LSH and Boosting SSC, a different retrieval algorithm is used whereby the binary code only creates a shortlist and exhaustive search is performed on the shortlist. Such an algorithm is impractical for the scale of data we are considering.

# References

[1] R. R. Salakhutdinov and G. E. Hinton. Learning a nonlinear embedding by preserving class neighbourhood structure. In *AISTATS*, 2007.

[2] A. Torralba, R. Fergus, and W. T. Freeman. Tiny images. Technical Report MIT-CSAIL-TR-2007-024, Computer Science and Artificial Intelligence Lab, Massachusetts Institute of Technology, 2007.

[3] A. Torralba, R. Fergus, and Y. Weiss. Small codes and large databases for recognition. In *CVPR*, 2008.

[4] James Hays and Alexei A Efros. Scene completion using millions of photographs. *ACM Transactions on Graphics (SIGGRAPH 2007)*, 26(3), 2007.

[5] R. R. Salakhutdinov and G. E. Hinton. Semantic hashing. In *SIGIR workshop on Information Retrieval and applications of Graphical Models*, 2007.

[6] Mikhail Belkin and Partha Niyogi. Laplacian eigenmaps and spectral techniques for embedding and clustering. In *NIPS*, pages 585–591, 2001.

[7] Geoffrey E. Hinton and Sam T. Roweis. Stochastic neighbor embedding. In *NIPS*, pages 833–840, 2002.

[8] A. Andoni and P. Indyk. Near-optimal hashing algorithms for approximate nearest neighbor in high dimensions. In *FOCS*, pages 459–468, 2006.

[9] R. R. Salakhutdinov and G. E. Hinton. Learning a nonlinear embedding by preserving class neighbourhood structure. In *AI and Statistics*, 2007.

[10] G. Shakhnarovich, P. Viola, and T. Darrell. Fast pose estimation with parameter sensitive hashing. In *ICCV*, 2003.

[11] A.Y. Ng, M.I. Jordan, and Y. Weiss. On spectral clustering, analysis and an algorithm. In *Advances in Neural Information Processing 14*, 2001.

[12] J. Shi and J. Malik. Normalized cuts and image segmentation. In *Proc. IEEE Conf. Computer Vision and Pattern Recognition*, pages 731–737, 1997.

[13] Yoshua Bengio, Olivier Delalleau, Nicolas Le Roux, Jean-François Paiement, Pascal Vincent, and Marie Ouimet. Learning eigenfunctions links spectral embedding and kernel pca. *Neural Computation*, 16(10):2197–2219, 2004.

[14] Charless Fowlkes, Serge Belongie, Fan R. K. Chung, and Jitendra Malik. Spectral grouping using the nyström method. *IEEE Trans. Pattern Anal. Mach. Intell.*, 26(2):214–225, 2004.

[15] R. R. Coifman, S. Lafon, A. B. Lee, M. Maggioni, B. Nadler, F. Warner, and S. W. Zucker. Geometric diffusions as a tool for harmonic analysis and structure definition of data: Diffusion maps. *Proceedings of the National Academy of Sciences*, 102(21):7426–7431, May 2005.

[16] M. Belkin and P. Niyogi. Towards a theoretical foundation for laplacian based manifold methods. *Journal of Computer and System Sciences*, 2007.

[17] Boaz Nadler, Stephane Lafon amd Ronald R. Coifman, and Ioannis G. Kevrekidis. Diffusion maps, spectral clustering and reaction coordinates of dynamical systems. *Arxiv*, 2008. http://arxiv.org/.

